# A Humanlike Predictor of Facial Attractiveness

**Amit Kagian**[*1], **Gideon Dror**[‡2], **Tommer Leyvand**[*3], **Daniel Cohen-Or**[*4], **Eytan Ruppin**[*5]

[*] School of Computer Sciences, Tel-Aviv University, Tel-Aviv, 69978, Israel.
[‡] The Academic College of Tel-Aviv-Yaffo, Tel-Aviv, 64044, Israel.

Email: {[1]kagianam, [3]tommer, [4]dcor, [5]ruppin}@post.tau.ac.il, [2]gideon@mta.ac.il

## Abstract

This work presents a method for estimating human facial attractiveness, based on supervised learning techniques. Numerous facial features that describe facial geometry, color and texture, combined with an average human attractiveness score for each facial image, are used to train various predictors. Facial attractiveness ratings produced by the final predictor are found to be highly correlated with human ratings, markedly improving previous machine learning achievements. Simulated psychophysical experiments with virtually manipulated images reveal preferences in the machine's judgments which are remarkably similar to those of humans. These experiments shed new light on existing theories of facial attractiveness such as the averageness, smoothness and symmetry hypotheses. It is intriguing to find that a machine trained explicitly to capture an operational performance criteria such as attractiveness rating, implicitly captures basic human psychophysical biases characterizing the perception of facial attractiveness in general.

## 1 Introduction

Philosophers, artists and scientists have been trying to capture the nature of beauty since the early days of philosophy. Although in modern days a common layman's notion is that judgments of beauty are a matter of subjective opinion, recent findings suggest that people might share a common taste for facial attractiveness and that their preferences may be an innate part of the primary constitution of our nature. Several experiments have shown that 2 to 8 months old infants prefer looking at faces which adults rate as being more attractive [1]. In addition, attractiveness ratings show very high agreement between groups of raters belonging to the same culture and even across cultures [2]. Such findings give rise to the quest for common factors which determine human facial attractiveness. Accordingly, various hypotheses, from cognitive, evolutionary and social perspectives, have been put forward to describe the common preferences for facial beauty.

Inspired by Sir Francis Galton's photographic method of composing faces [3], Rubenstein, Langlois and Roggman created averaged faces by morphing multiple images together and proposed that averageness is the answer for facial attractiveness [4, 5]. Human judges found these averaged faces to be attractive and rated them with attractiveness ratings higher than the mean rating of the component faces composing them. Grammer and Thornhill have investigated symmetry and averageness of faces and concluded that symmetry was more important than averageness in facial attractiveness [6]. Little and colleagues have agreed that average faces are attractive but claim that faces with certain extreme features, such as extreme sexually dimorphic traits, may be more attractive than average faces [7]. Other

researchers have suggested various conditions which may contribute to facial attractiveness such as neonate features, pleasant expressions and familiarity. Cunningham and his associates suggest a multiple fitness model in which there is no single constructing line that determines attractiveness. Instead, different categories of features signal different desirable qualities of the perceived target [8]. Even so, the multiple fitness model agrees that some facial qualities are universally physically attractive to people.

Apart from eliciting the facial characteristics which account for attractiveness, modern researchers try to describe underlying mechanisms for these preferences. Many contributors refer to the evolutionary origins of attractiveness preferences [9]-[11]. According to this view, facial traits signal mate quality and imply chances for reproductive success and parasite resistance. Some evolutionary theorists suggest that preferred features might not signal mate quality but that the "good taste" by itself is an evolutionary adaptation (individuals with a preference for attractiveness will have attractive offspring that will be favored as mates) [9]. Another mechanism explains attractiveness' preferences through a cognitive theory - a preference for attractive faces might be induced as a by-product of general perception or recognition mechanisms [5, 12]: Attractive faces might be pleasant to look at since they are closer to the cognitive representation of the face category in the mind. These cognitive representations are described as a part of a cognitive mechanism that abstracts prototypes from distinct classes of objects. These prototypes relate to average faces when considering the averageness hypothesis. A third view has suggested that facial attractiveness originates in a social mechanism, where preferences may be dependent on the learning history of the individual and even on his social goals [12].

Different studies have tried to use computational methods in order to analyze facial attractiveness. Averaging faces with morph tools was done in several cases (e.g. [5, 13]). In [14], laser scans of faces were put into complete correspondence with the average face in order to examine the relationship between facial attractiveness, age, and averageness. Another approach was used in [15] where a genetic algorithm, guided by interactive user selections, was programmed to evolve a "most beautiful" female face. [16] used machine learning methods to investigate whether a machine can predict attractiveness ratings by learning a mapping from facial images to their attractiveness scores. Their predictor achieved a significant correlation of 0.6 with average human ratings, demonstrating that facial beauty can be learned by a machine, at least to some degree. However, as human raters still do significantly outperform the predictor of [16], the challenge of constructing a facial attractiveness machine with human level evaluation accuracy has remained open. A primary goal of this study is to surpass these results by **developing a machine which obtains human level performance in predicting facial attractiveness.** Having accomplished this, our second main goal is to conduct a series of simulated psychophysical experiments and study the resemblance between human and machine judgments. This latter task carries two potential rewards: A. To determine whether the machine can aid in understanding the psychophysics of human facial attractiveness, capitalizing on the ready accessibility of the analysis of its inner workings, and B. **To study whether learning an explicit operational ratings prediction task also entails learning implicit humanlike biases**, at least for the case of facial attractiveness.

## 2   The facial training database: Acquisition, preprocessing and representation

### 2.1   Rating facial attractiveness

The chosen database was composed of 91 facial images of American females, taken by the Japanese photographer Akira Gomi. All 91 samples were frontal color photographs of young Caucasian females with a neutral expression. All samples were of similar age, skin color and gender. The subjects' portraits had no accessories or other distracting items such as jewelry. All 91 facial images in the dataset were rated for attractiveness by 28 human raters (15 males, 13 females) on a 7-point Likert scale (1 = very unattractive, 7 = very attractive). Ratings were collected with a specifically designed html interface. Each rater was asked to

view the entire set before rating in order to acquire a notion of attractiveness scale. There was no time limit for judging the attractiveness of each sample and raters could go back and adjust the ratings of already rated samples. The images were presented to each rater in a random order and each image was presented on a separate page. The final attractiveness rating of each sample was its mean rating across all raters. To validate that the number of ratings collected adequately represented the ``collective attractiveness rating'' we randomly divided the raters into two disjoint groups of equal size. For each facial image, we calculated the mean rating on each group, and calculated the Pearson correlation between the mean ratings of the two groups. This process was repeated 1,000 times. The mean correlation between two groups was 0.92 ($\sigma = 0.01$). This corresponds well to the known level of consistency among groups of raters reported in the literature (e.g. [2]). Hence, the mean ratings collected are stable indicators of attractiveness that can be used for the learning task. The facial set contained faces in all ranges of attractiveness. Final attractiveness ratings range from 1.42 to 5.75 and the mean rating was 3.33 ($\sigma = 0.94$).

## 2.2   Data preprocessing and representation

Preliminary experimentation with various ways of representing a facial image have systematically shown that features based on measured proportions, distances and angles of faces are most effective in capturing the notion of facial attractiveness (e.g. [16]). To extract facial features we developed an automatic engine that is capable of identifying eyes, nose, lips, eyebrows, and head contour. In total, we measured 84 coordinates describing the locations of those facial features (Figure 1). Several regions are suggested for extracting mean hair color, mean skin color and skin texture. The feature extraction process was basically automatic but some coordinates needed to be manually adjusted in some of the images. The facial coordinates are used to create a *distances-vector* of all 3,486 distances between all pairs of coordinates in the complete graph created by all coordinates. For each image, all distances are normalized by face length. In a similar manner, a *slopes-vector* of all the 3,486 slopes of the lines connecting the facial coordinates is computed. Central fluctuating asymmetry (CFA), which is described in [6], is calculated from the coordinates as well. The application also provides, for each face, Hue, Saturation and Value (HSV) values of hair color and skin color, and a measurement of skin smoothness.

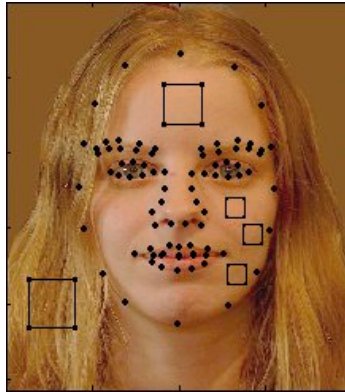

Figure 1: Facial coordinates with hair and skin sample regions as represented by the facial feature extractor. Coordinates are used for calculating geometric features and asymmetry. Sample regions are used for calculating color values and smoothness. The sample image, used for illustration only, is of T.G. and is presented with her full consent.

Combining the distances-vector and the slopes-vector yields a vector representation of 6,972 *geometric features* for each image. Since strong correlations are expected among the features in such representation, principal component analysis (PCA) was applied to these geometric features, producing 90 principal components which span the sub-space defined by the 91 image vector representations. The geometric features are projected on those 90 principal components and supply 90 orthogonal *eigenfeatures* representing the geometric features. Eight measured features were not included in the PCA analysis, including CFA, smoothness, hair color coordinates (HSV) and skin color coordinates. These features are

assumed to be directly connected to human perception of facial attractiveness and are hence kept at their original values. These 8 features were added to the 90 geometric eigenfeatures, resulting in a total of 98 *image-features* representing each facial image in the dataset.

## 3   Experiments and results

### 3.1   Predictor construction and validation

We experimented with several induction algorithms including simple Linear Regression, Least Squares Support Vector Machine (LS-SVM) (both linear as well as non-linear) and Gaussian Processes (GP). However, as the LS-SVM and GP showed no substantial advantage over Linear Regression, the latter was used and is presented in the sequel.

A key ingredient in our methods is to use a proper image-features selection strategy. To this end we used subset feature selection, implemented by ranking the image-features by their Pearson correlation with the target. Other ranking functions produced no substantial gain. To measure the performance of our method we removed one sample from the whole dataset. This sample served as a test set. We found, for each left out sample, the optimal number of image-features by performing leave-one-out-cross-validation (LOOCV) on the remaining samples and selecting the number of features that minimizes the absolute difference between the algorithm's output and the targets of the training set. In other words, the score for a test example was predicted using a single model based on the training set only. This process was repeated n=91 times, once for each image sample. The vector of attractiveness predictions of all images is then compared with the true targets. These scores are found to be in a high Pearson correlation of 0.82 with the mean ratings of humans (*P-value* $< 10^{-23}$), which corresponds to a normalized Mean Squared Error of 0.39. This accuracy is a marked improvement over the recently published performance results of a Pearson correlation of 0.6 on a similar dataset [16]. The average correlation of an individual human rater to the mean correlations of all other raters in our dataset is 0.67 and the average correlation between the mean ratings of groups of raters is 0.92 (section 2.1).

It should be noted that we tried to use this feature selection and training procedure with the original geometric features instead of the eigenfeatures, ranking them by their correlation to the targets and selecting up to 300 best ranked features. This, however, has failed to produce good predictors due to strong correlations between the original geometric features (maximal Pearson correlation obtained was 0.26).

### 3.2   Similarity of machine and human judgments

Each rater (human and machine) has a 91 dimensional *rating vector* describing its

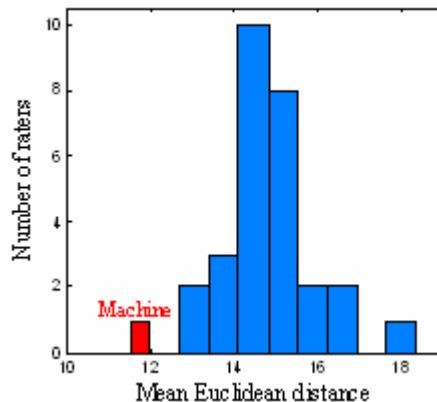

Figure 2: Distribution of mean Euclidean distance from each human rater to all other raters in the *ratings space*. The machine's average distance form all other raters (left bar) is smaller than the average distance of each of the human raters to all others.

attractiveness ratings of all 91 images. These vectors can be embedded in a 91 dimensional *ratings space*. The Euclidian distance between all raters (human and machine) in this space was computed. Compared with each of the human raters, the ratings of the machine were the closest, on average, to the ratings of all other human raters (Figure 2). To verify that the machine ratings are not outliers that fall out of clusters of human raters (even though their mean distance from the other ratings is small) we surrounded each of the rating vectors in the ratings space with multidimensional spheres of several radius sizes. The machine had more human neighbors than the mean number of neighbors of human raters, testifying that it does not fall between clusters. Finally, for a graphic display of machine ratings among human ratings we applied PCA to machine and human ratings in the rating space and projected all ratings onto the resulting first 2 and 3 principal components. Indeed, the machine is well placed in a mid-zone of human raters (Figure 3).

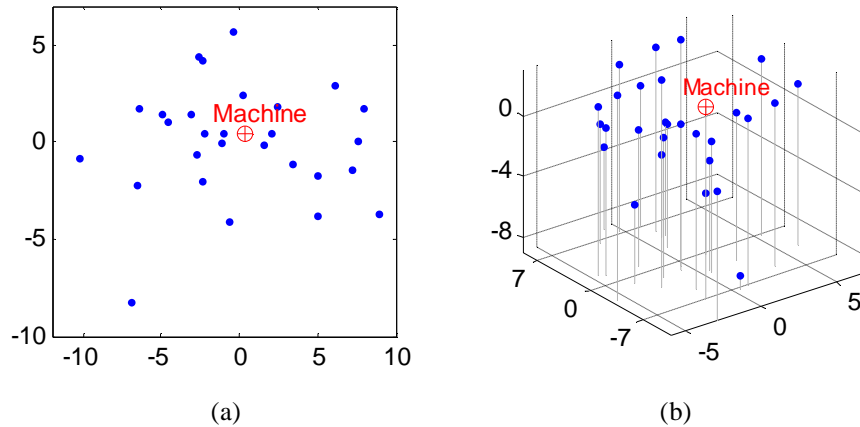

(a)                                    (b)

Figure 3: Location of machine ratings among the 28 human ratings: Ratings were projected into 2 dimensions (a) and 3 dimensions (b) by performing PCA on all ratings and projecting them on the first principal components. The projected data explain 29.8% of the variance in (a) and 36.6% in (b).

### 3.3 Psychophysical experiments in silico

A number of simulated psychophysical experiments reveal humanlike biases of the machine's performance. Rubenstein et al. discuss a morphing technique to create mathematically averaged faces from multiple face images [5]. They reported that averaged faces made of 16 and 32 original component images were rated higher in attractiveness than the mean attractiveness ratings of their component faces and higher than composites consisting of fewer faces. In their experiment, 32-component composites were found to be the most attractive. We used a similar technique to create averaged virtually-morphed faces with various numbers of components, $n_c$, and have let the machine predict their attractiveness. To this end, coordinate values of the original component faces were averaged to create a new set of coordinates for the composite. These coordinates were used to calculate the geometrical features and CFA of the averaged face. Smoothness and HSV values for the composite faces were calculated by averaging the corresponding values of the component faces[1]. To study the effect of $n_c$ on the attractiveness score we produced 1,000 virtual morph images for each value of $n_c$ between 2 and 50, and used our attractiveness predictor (section 3.1) to compute the attractiveness scores of the resulting composites.

In accordance with the experimental results of [5], the machine manifests a humanlike bias for higher scores of averaged composites over their components' mean score. Figure 4a, presenting these results, shows the percent of components which were rated as less attractive than their corresponding composite, for each number of components $n_c$. As evident, the attractiveness rating of a composite surpasses a larger percent of its components' ratings as $n_c$ increases. Figure 4a also shows the mean scores of 1,000

composites and the mean scores of their components, for each $n_c$ (scores are normalized to the range [0, 1]). Their actual attractiveness scores are reported in Table 1. As expected, the mean scores of the components images are independent of $n_c$, while composites' scores increase with $n_c$. Mean values of smoothness and asymmetry of the composites are presented in Figure 4b.

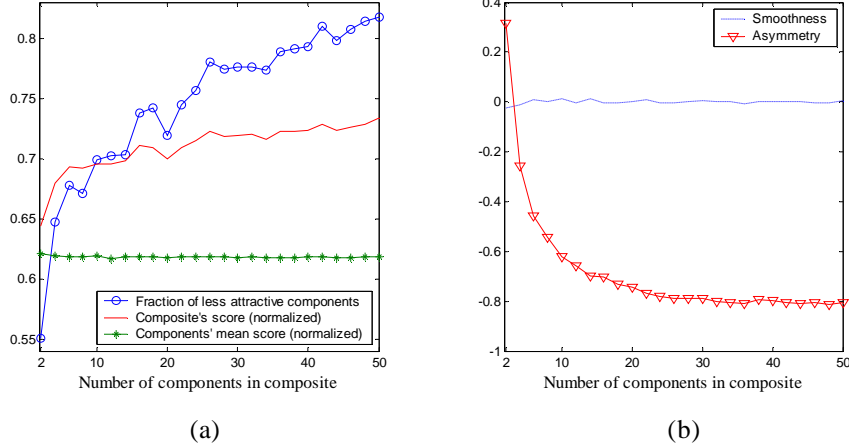

(a)            (b)

Figure 4: Mean results over 1,000 composites made of varying numbers of image components: (a) Percent of components which were rated as less attractive than their corresponding composite accompanied with mean scores of composites and the mean scores of their components (scores are normalized to the range [0, 1]. actual attractiveness scores are reported in Table 1). (b) Mean values of smoothness and asymmetry of 1,000 composites for each number of components, $n_c$.

Table 1: Mean results over 1,000 composites made of varying numbers of component images

| NUMBER OF COMPONENTS IN COMPOSITE | COMPOSITE SCORE | COMPONENTS MEAN SCORE | COMPONENTS RATED LOWER THAN COMPOSITE (PERCENT) |
|---|---|---|---|
| 2 | 3.46 | 3.34 | 55 % |
| 4 | 3.66 | 3.33 | 64 % |
| 12 | 3.74 | 3.32 | 70 % |
| 25 | 3.82 | 3.32 | 75 % |
| 50 | 3.94 | 3.33 | 81 % |

Recent studies have provided evidence that skin texture influences judgments of facial attractiveness [17]. Since blurring and smoothing of faces occur when faces are averaged together [5], the smooth complexion of composites may underlie the attractiveness of averaged composites. In our experiment, a preference for averageness is found even though our method of virtual-morphing does not produce the smoothening effect and the mean smoothness value of composites corresponds to the mean smoothness value in the original dataset, for all $n_c$ (see Figure 4b). Researchers have also suggested that averaged faces are attractive since they are exceptionally symmetric [18]. Figure 4b shows that the mean level of asymmetry is indeed highly correlated with the mean scores of the morphs (Pearson correlation of -0.91, *P-value* $< 10^{-19}$). However, examining the correlation between the rest of the features and the composites' scores reveals that this high correlation is not at all unique to asymmetry. In fact, 45 of the 98 features are strongly correlated with attractiveness scores (|Pearson correlation| $> 0.9$). The high correlation between these numerous features and attractiveness scores of averaged faces indicates that symmetry level is not an exceptional factor in the machine's preference for averaged faces. Instead, it suggests that

averaging causes many features, including both geometric features and symmetry, to change in a direction which causes an increase in attractiveness.

It has been argued that although averaged faces are found to be attractive, very attractive faces are not average [18]. A virtual composite made of the 12 most attractive faces in the set (as rated by humans) was rated by the machine with a high score of 5.6 while 1,000 composites made of 50 faces got a maximum score of only 5.3. This type of preference resembles the findings of an experiment by Perrett et al. in which a highly attractive composite, morphed from only attractive faces, was preferred by humans over a composite made of 60 images of all levels of attractiveness [13].

Another study by Zaidel et al. examined the asymmetry of attractiveness perception and offered a relationship between facial attractiveness and hemispheric specialization [19]. In this research right-right and left-left chimeric composites were created by attaching each half of the face to its mirror image. Subjects were asked to look at left-left and right-right composites of the same image and judge which one is more attractive. For women's faces, right-right composites got twice as many 'more attractive' responses than left-left composites. Interestingly, similar results were found when simulating the same experiment with the machine: Right-right and left-left chimeric composites were created from the extracted coordinates of each image and the machine was used to predict their attractiveness ratings (taking care to exclude the original image used for the chimeric composition from the training set, as it contains many features which are identical to those of the composite). The machine gave 63 out of 91 right-right composites a higher rating than their matching left-left composite, while only 28 left-left composites were judged as more attractive. A paired t-test shows these results to be statistically significant with $P\text{-value} < 10^{-7}$ (scores of chimeric composites are normally distributed). It is interesting to see that the machine manifests the same kind of asymmetry bias reported by Zaidel et al, though it has never been explicitly trained for that.

## 4  Discussion

In this work we produced a high quality training set for learning facial attractiveness of human faces. Using supervised learning methodologies we were able to construct the first predictor that achieves accurate, humanlike performance for this task. Our results add the task of facial attractiveness prediction to a collection of abstract tasks that has been successfully accomplished with current machine learning techniques.

Examining the machine and human raters' representations in the ratings space identifies the ratings of the machine in the center of human raters, and closest, in average, to other human raters. The similarity between human and machine preferences has prompted us to further study the machine's operation in order to capitalize on the accessibility of its inner workings and learn more about human perception of facial attractiveness. To this end, we have found that that the machine favors averaged faces made of several component faces. While this preference is known to be common to humans as well, researchers have previously offered different reasons for favoring averageness. Our analysis has revealed that symmetry is strongly related to the attractiveness of averaged faces, but is definitely not the only factor in the equation since about half of the image-features relate to the ratings of averaged composites in a similar manner as the symmetry measure. This suggests that a general movement of features toward attractiveness, rather than a simple increase in symmetry, is responsible for the attractiveness of averaged faces. Obviously, strictly speaking this can be held true only for the machine, but, in due of the remarkable ``humnalike'' behavior of the machine, it also brings important support to the idea that this finding may well extend also to human perception of facial attractiveness. Overall, it is quite surprising and pleasing to see that a machine trained explicitly to capture an operational performance criteria such as rating, implicitly captures basic human psychophysical biases related to facial attractiveness. It is likely that while the machine learns the ratings in an explicit supervised manner, it also concomitantly and implicitly learns other basic characteristics of human facial ratings, as revealed by studying its "psychophysics".

## Acknowledgments

We thank Dr. Bernhard Fink and the Ludwig-Boltzmann Institute for Urban Ethology at the Institute for Anthropology, University of Vienna, Austria, and Prof. Alice J. O'Toole from the University of Texas at Dallas, for kindly letting us use their face databases.

## Footnotes

[1] HSV values are converted to RGB before averaging

## References

[1] Langlois, J.H., Roggman, L.A., Casey, R.J., Ritter, J.M., Rieser-Danner, L.A. & Jenkins, V.Y. (1987) Infant preferences for attractive faces: Rudiments of a stereotype? *Developmental Psychology, 23*, 363-369.

[2] Cunningham, M.R., Roberts, A.R., Wu, C.-H., Barbee, A.P. & Druen, P.B. (1995) Their ideas of beauty are, on the whole, the same as ours: Consistency and variability in the cross-cultural perception of female physical attractiveness. *Journal of Personality and Social Psychology, 68*, 261-279.

[3] Galton, F. (1878) Composite portraits. *Journal of the Anthropological Institute of Great Britain and Ireland, 8*, 132-142.

[4] Langlois, J.H. & Roggman, L.A. (1990) Attractive faces are only average. *Psychological Science, 1*, 115-121.

[5] Rubenstein, A.J., Langlois, J.H & Roggman, L.A. (2002) What makes a face attractive and why: The role of averageness in defining facial beauty. In Rhodes, G. & Zebrowitz, L.A. (eds.), *Advances in Visual Cognition, Vol. 1: Facial Attractiveness,* pp. 1-33. Westport, CT: Ablex.

[6] Grammer, K. & Thornhill, R. (1994) Human (*Homo sapiens*) facial attrativness and sexual selection: The role of symmetry and averageness. *Journal of Comparative Psychology, 108*, 233-242.

[7] Little, A.C., Penton-Voak, I.S., Burt, D.M. & Perrett, D.I. (2002) Evolution and individual differences in the perception of attractiveness: How cyclic hormonal changes and self-perceived attractiveness influence female preferences for male faces. In Rhodes, G. & Zebrowitz, L.A. (eds.), *Advances in Visual Cognition, Vol. 1: Facial Attractiveness,* pp. 59-90. Westport, CT: Ablex.

[8] Cunningham, M.R., Barbee, A.P. & Philhower, C.L. (2002) Dimensions of facial physical attractiveness: The intersection of biology and culture. In Rhodes, G. & Zebrowitz, L.A. (eds.), *Advances in Visual Cognition, Vol. 1: Facial Attractiveness,* pp. 193-238. Westport, CT: Ablex.

[9] Thornhill, R. & Gangsted, S.W. (1999) Facial Attractiveness. *Trends in Cognitive Sciences, 3*, 452-460.

[10] Andersson, M. (1994) *Sexual Selection*. Princeton, NJ: Princeton University Press.

[11] Møller, A.P. & Swaddle, J.P. (1997) *Asymmetry, developmental stability, and evolution*. Oxford: Oxford University Press.

[12] Zebrowitz, L.A. & Rhodes, G. (2002) Nature let a hundred flowers bloom: The multiple ways and wherefores of attractiveness. In Rhodes, G. & Zebrowitz, L.A. (eds.), *Advances in Visual Cognition, Vol. 1: Facial Attractiveness,* pp. 261-293. Westport, CT: Ablex.

[13] Perrett, D.I., May, K.A. & Yoshikawa, S. (1994) facial shape and judgments of female attractiveness. *Nature, 368*, 239-242.

[14] O´Toole, A.J., Price, T., Vetter, T., Bartlett, J.C. & Blanz, V. (1999) 3D shape and 2D surface textures of human faces: the role of "averages" in attractiveness and age. *Image and Vision Computing, 18*, 9-19.

[15] Johnston, V. S. & Franklin, M. (1993) Is beauty in the eye of the beholder? *Ethology and Sociobiology, 14*, 183-199.

[16] Eisenthal, Y., Dror, G. & Ruppin, E. (2006) Facial attractiveness: Beauty and the Machine. *Neural Computation, 18*, 119-142.

[17] Fink, B., Grammer, K. & Thornhill, R. (2001) Human (Homo sapiens) Facial Attractiveness in Relation to Skin Texture and Color. *Journal of Comparative Psychology*, 115, 92–99.

[18] Alley, T.R. & Cunningham, M.R. (1991) Averaged faces are attractive but very attractive faces are not average. *Psychological Science, 2*, 123-125.

[19] Zaidel, D.W., Chen, A.C. & German, C. (1995) She is not a beauty even when she smiles: possible evolutionary basis for a relationship between facial attractiveness and hemispheric specialization. *Neuropsychologia, 33(5)*, 649-655
